# Log-Linear Models for Label Ranking

**Ofer Dekel**
Computer Science & Eng.
Hebrew University
oferd@cs.huji.ac.il

**Christopher D. Manning**
Computer Science Dept.
Stanford University
manning@cs.stanford.edu

**Yoram Singer**
Computer Science & Eng.
Hebrew University
singer@cs.huji.ac.il

## Abstract

Label ranking is the task of inferring a total order over a predefined set of labels for each given instance. We present a general framework for batch learning of label ranking functions from supervised data. We assume that each instance in the training data is associated with a list of preferences over the label-set, however we do not assume that this list is either complete or consistent. This enables us to accommodate a variety of ranking problems. In contrast to the general form of the supervision, our goal is to learn a ranking function that induces a *total* order over the *entire* set of labels. Special cases of our setting are multilabel categorization and hierarchical classification. We present a general boosting-based learning algorithm for the label ranking problem and prove a lower bound on the progress of each boosting iteration. The applicability of our approach is demonstrated with a set of experiments on a large-scale text corpus.

## 1  Introduction

This paper discusses supervised learning of label rankings – the task of associating instances with a total order over a predefined set of labels. The ordering should be performed in accordance with some notion of relevance of the labels. That is, a label deemed relevant to an instance should be ranked higher than a label which is considered less relevant. With each training instance we receive supervision given as a set of preferences over the labels. Concretely, the supervision we receive with each instance is given in the form of a *preference graph*: a simple directed graph for which the labels are the graph vertices. A directed edge from a label $y$ to another label $y'$ denotes that according to the supervision, $y$ is more relevant to the instance than $y'$. We do not impose any further constraints on the structure of the preference graph.

The approach we employ distills and generalizes several learning settings. The simplest setting is multiclass categorization in which each instance is associated with a *single* label out of $k$ possible labels. Such a setting was discussed for instance in [10] where a boosting algorithm called AdaBoost.MR (MR stands for Multiclass Ranking) for solving this problem was described and analyzed. Using the graph representation for multiclass problems, the preference graph induced by the supervision has $k$ vertices and $k-1$ edges. A directed edge points from the (single) relevant label to each of the $k-1$ irrelevant labels (Fig. 1a). An interesting and practical generalization of multiclass problems is multilabel problems [10, 6, 4], in which a set of relevant labels (rather than a single label) is associated with each instance. In this case the supervision is represented by a directed *bipartite*

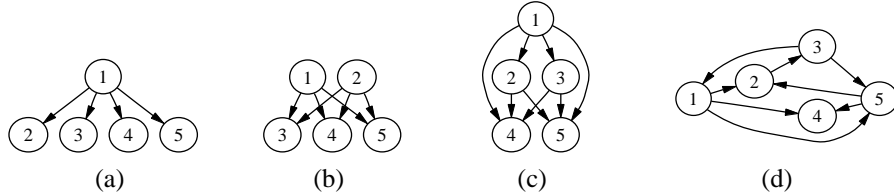

$$(a) \qquad\qquad (b) \qquad\qquad (c) \qquad\qquad (d)$$

Figure 1: The supervision provided to the algorithm associates every training instance with a *preference graph*. Different graph topologies define different learning problems. Examples that fit naturally in our generalized setting: (a) multiclass single-label categorization where 1 is the correct label. (b) multiclass multilabel categorization where $\{1, 2\}$ is the set of correct labels. (c) A multi-layer graph that encodes three levels of label "goodness", useful for instance in hierarchical multiclass settings. (d) a general (possibly cyclic) preference graph with no predefined structure.

graph where the relevant labels constitute one side of the graph and the irrelevant labels the other side and there is a directed edge from each relevant label to each irrelevant label. (Fig. 1b). Similar settings are also encountered in information retrieval and language processing tasks. In these settings the set of labels contains linguistic structures such as tags and parses [1, 12] and the goal is to produce a total order over, for instance, candidate parses. The supervision might consist of information that distinguishes three goodness levels (Fig. 1c); for instance, the Penn Treebank [13] has notations to mark not only the most likely correct parse implicitly opposed to incorrect parses, but also to mark other possibly correct parses involving different phrasal attachments (additional information that almost all previous work in parsing has ignored). Additionally, one can more fully rank the quality of the many candidate parses generated for a sentence based on how many constituents or dependencies each shares with the correct parse – much more directly and effectively approaching the metrics on which parser quality is usually assessed. For concreteness, we use the term label ranking for all of these problems.

Our learning framework decomposes each preference graph into subgraphs, where the graph decomposition procedure may take a general form and can change as a function of the instances. Ranking algorithms, especially in multilabel categorization problems, often reduce the ranking task into multiple binary decision problems by enumerating over all pairs of labels [7, 6, 4]. Such a reduction can easily be accommodated within our framework by decomposing the preference graph into elementary subgraphs, each consisting of a single edge. Another approach is to compare a highly preferred label (such as the correct or best parse of a sentence) with less preferred labels. Such approaches can be analyzed within our framework by defining a graph decomposition procedure that generates a subgraph for each relevant label and the neighboring labels that it is preferred over. Returning to multilabel settings, this decomposition amounts to a loss that counts the number of relevant labels which are wrongly ranked below irrelevant ones.

The algorithmic core of this paper is based on boosting-style algorithms for exponential models [2, 8]. Specifically, the boosting-style updates we employ build upon the construction used in [2] for solving multiclass problems. Our framework employing graph decomposition can also be used in other settings such as element ranking via projections [3, 11]. Furthermore, settings in which a semi-metric is defined over the label-set can also be reduced to the problem of label ranking, such as the parse ordering case mentioned above or when the labels are arranged in a hierarchical structure. We employ such a reduction in the category ranking experiments described in Sec. 4.

The paper is organized as follows: a formal description of our setting is given in Sec. 2. In Sec. 3 we present an algorithm for learning label ranking functions. We demonstrate the merits of our approach on the task of category-ranking in Sec. 4 and conclude in Sec. 5.

## 2   Problem Setting

Let $\mathcal{X}$ be an instance domain and let $\mathcal{Y}$ be a set of labels, possibly of infinite cardinality. A *label ranking* for an instance $\mathbf{x} \in \mathcal{X}$ is a total order over $\mathcal{Y}$, where $y \succ y'$ implies that $y$ is preferred over $y'$ as a label for $\mathbf{x}$. A *label ranking function* $f : \mathcal{X} \times \mathcal{Y} \rightarrow \mathbb{R}$ induces a label ranking for $\mathbf{x} \in \mathcal{X}$ by $y \succ y' \iff f(\mathbf{x}, y) > f(\mathbf{x}, y')$. Overloading our notation, we denote the label ranking induced by $f$ for $\mathbf{x}$ by $f(\mathbf{x})$.

We assume that we are provided with a set of base label-ranking functions, $h_1, \ldots, h_n$, and aim to learn a linear combination of the form $f(x, y) = \sum_{j=1}^{n} \lambda_j h_j(x, y)$. We are also provided with a training set $S = \{(\mathbf{x}_i, \mathbf{G}_i)\}_{i=1}^{m}$ where every example is comprised of an instance $\mathbf{x}_i \in \mathcal{X}$ and a preference graph $\mathbf{G}_i$. As defined in the previous section, a preference graph is a directed graph $G = (V, E)$, for which the set of vertices $V$ is defined to be the set of labels $\mathcal{Y}$ and $E$ is some finite set of directed edges. Every edge in a directed graph $e \in E$ is associated with an *initial vertex*, $\text{init}(e) \in V$, and a *terminal vertex*, $\text{term}(e) \in V$. The existence of a directed edge between two labels in a preference graph indicates that $\text{init}(e)$ is preferred over $\text{term}(e)$ and should be ranked higher. We require preference graphs to be simple, namely to have no more than a single edge between any pair of vertices and to not contain any self-loops. However, we impose no additional constraints on the supervision, namely, the set of edges in a preference graph may be sparse and may even include cycles. This form of supervision was chosen for its generality and flexibility. If $\mathcal{Y}$ is very large (possibly infinite), it would be unreasonable to require that the training data contain a complete total order over $\mathcal{Y}$ for every instance.

Informally, our goal is for the label ranking induced by $f$ to be as consistent as possible with all of the preference graphs given in $S$. We say that $f(\mathbf{x}_i)$ disagrees with a preference graph $\mathbf{G}_i = (V_i, E_i)$ if there exists an edge $e \in E_i$ for which $f(\mathbf{x}_i, \text{init}(e)) \leq f(\mathbf{x}_i, \text{term}(e))$. Formally, we define a function $\delta$ that indicates when such a disagreement occurs

$$\delta(f(\mathbf{x}), \mathbf{G}) = \begin{cases} 1 & \text{if } \exists e \in E \text{ s.t. } f(\mathbf{x}, \text{init}(e)) \leq f(\mathbf{x}, \text{term}(e)) \\ 0 & \text{otherwise} \end{cases} .$$

A simple measure of empirical ranking accuracy immediately follows from the definition of $\delta$: We define the $0 - 1$ *error* attained by a ranking function $f$ on a training set $S$ to be the number of training examples for which $f(\mathbf{x}_i)$ disagrees with $\mathbf{G}_i$, namely,

$$\varepsilon_{0-1}(f, S) = \sum_{i=1}^{m} \delta(f(\mathbf{x}_i), \mathbf{G}_i) .$$

The $0 - 1$ error may be natural for certain ranking problems, however in general it is a rather crude measure of ranking inaccuracy, as it is invariant to the exact number of edges in $\mathbf{G}_i$ with which $f(\mathbf{x}_i)$ disagrees. Many ranking problems require a more refined notion of ranking accuracy. Thus, we define the *disagreement error* attained by $f(\mathbf{x}_i)$ with respect to $\mathbf{G}_i$ to be the fraction of edges in $E_i$ with which $f(\mathbf{x}_i)$ disagrees. The disagreement error attained on the entire training set is the sum of disagreement errors over all training examples. Formally, we define the disagreement error attained on $S$ as

$$\varepsilon_{\text{dis}}(f, S) = \sum_{i=1}^{m} \frac{\left| \left\{ e \in E_i \text{ s.t. } f(\mathbf{x}, \text{init}(e)) \leq f(\mathbf{x}, \text{term}(e)) \right\} \right|}{|E_i|} .$$

Both the $0 - 1$ error and the disagreement error are reasonable measures of ranking inaccuracy. It turns out that both are instances of a more general notion of ranking error of which additional meaningful instances exist. The definition of this generalized error is slightly more involved but enables us to present a unified account of different measures of error.

The missing ingredient needed to define the generalized error is a graph decomposition procedure $\mathcal{A}$ that we assume is given together with the training data. $\mathcal{A}$ takes as its input

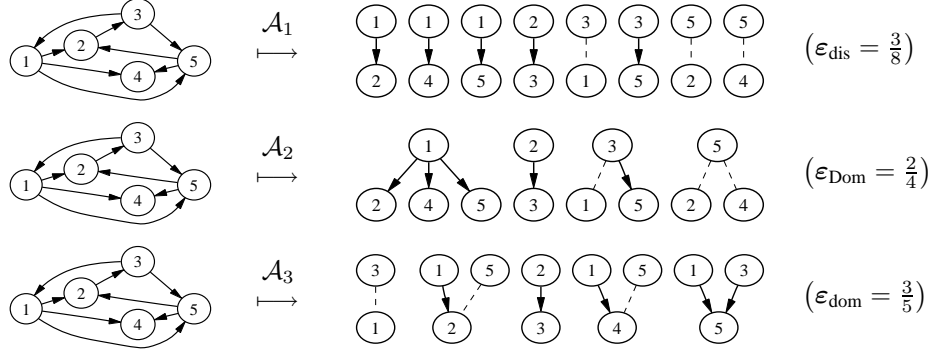

Figure 2: Applying different graph decomposition procedures induces different error functions: $\mathcal{A}_1$ induces $\varepsilon_{\mathrm{dis}}$, $\mathcal{A}_2$ induces $\varepsilon_{\mathrm{Dom}}$ and $\mathcal{A}_3$ induces $\varepsilon_{\mathrm{dom}}$. The errors above are with respect to the order $1 \succ 2 \succ 3 \succ 4 \succ 5$. Dashed edges without arrowheads disagree with this total order, and the errors are the fraction of subgraphs that contain disagreeing edges.

a preference graph $\mathbf{G}_i$ and returns a set of $s_i$ subgraphs of $\mathbf{G}_i$, denoted $\{G_{i,1}, \ldots, G_{i,s_i}\}$, where $G_{i,k} = (V_i, E_{i,k})$. Each subgraph $G_{i,k}$ is itself a preference graph and therefore $\delta(f(\mathbf{x}_i), G_{i,k})$ is well defined. We now define the *generalized error* attained by $f(\mathbf{x}_i)$ with respect to $\mathbf{G}_i$ as the fraction of subgraphs in $\mathcal{A}(\mathbf{G}_i)$ with which $f(\mathbf{x}_i)$ disagrees. The generalized error attained on $S$ is the sum of generalized errors over all training instances. Formally, the generalized ranking error is defined as

$$\varepsilon_{\mathrm{gen}}(f, S, \mathcal{A}) = \sum_{i=1}^{m} \frac{1}{s_i} \sum_{k=1}^{s_i} \delta(f(\mathbf{x}_i), G_{i,k}) \quad \text{where} \ \{G_{i,1}, \ldots, G_{i,s_i}\} = \mathcal{A}(\mathbf{G}_i) \ . \quad (1)$$

Previously used losses for label ranking are special cases of the generalized error and are derived by choosing an appropriate decomposition procedure $\mathcal{A}$. For instance, when $\mathcal{A}$ is defined to be the identity transformation on graphs ($\mathcal{A}(\mathbf{G}) = \{\mathbf{G}\}$), then the generalized ranking error is reduced to the $0-1$ error. Alternatively, for a graph $\mathbf{G}$ with $s$ edges, we can define $\mathcal{A}$ to return $s$ different subgraphs of $\mathbf{G}$, each consisting of a single edge from $\mathbf{G}$ (Fig. 2 top) and the generalized ranking error reduces to the disagreement error.

An additional meaningful measure of error is the *domination error*. A vertex is said to dominate the set of neighboring vertices that are connected to its outgoing edges. We would like every vertex in the preference graph to be ranked above all of its dominated neighbors. The domination error attained by $f(\mathbf{x}_i)$ with respect to $\mathbf{G}_i$ is the fraction of vertices with outgoing edges which are not ranked above all of their dominated neighbors. Formally, let $\mathcal{A}$ be the procedure that takes a preference graph $\mathbf{G} = (V, E)$ and returns a subgraph for each vertex with outgoing edges, each such subgraph consisting of a dominating vertex, its dominated neighbors and edges between them (Fig. 2 middle). Now define $\varepsilon_{\mathrm{Dom}}(f, S) = \varepsilon_{\mathrm{gen}}(f, S, \mathcal{A})$ . Minimizing the domination error is useful for solving multilabel classification problems. In these problems $\mathcal{Y}$ is of finite cardinality and every instance $\mathbf{x}_i$ is associated with a set of correct labels $Y_i \subseteq \mathcal{Y}$. In order to reduce this problem to a ranking problem, we construct preference graphs $G_i = (\mathcal{Y}, E_i)$, where $E_i$ contains edges from every vertex in $Y_i$ to every vertex in $\mathcal{Y} \setminus Y_i$. In this case, the domination loss simply counts the number of labels in $Y_i$ that are not ranked above all of the labels in $\mathcal{Y} \setminus Y_i$.

A final interesting measure of error is the *dominated error*, denoted $\varepsilon_{\mathrm{dom}}$. The dominated error is proportional to the number of labels with incoming edges that are not ranked below all of the labels that dominate them. Its graph decomposition procedure is depicted at the bottom of Fig. 2. Additional instances of the generalized ranking error exist, and can be tailored to fit most ranking problems. In the next section we set aside the specifics of the decomposition procedure and derive a minimization procedure for the generalized error.

INPUT: training data $S = \{(\mathbf{x}_i, \mathbf{G}_i)\}_{i=1}^{m}$ s.t. $\mathbf{x}_i \in \mathcal{X}$ and $\mathbf{G}_i$ is a preference graph,
a decomposition procedure $\mathcal{A}$ and a set of base ranking functions $\{h_1, \ldots, h_n\}$.

INITIALIZE: $\boldsymbol{\lambda}_1 = (0, 0, \ldots, 0)$

$$\pi_{i,e,j} = h_j\big(\mathbf{x}_i, \text{term}(e)\big) - h_j\big(\mathbf{x}_i, \text{init}(e)\big) \qquad [1 \le i \le m,\ e \in E_i,\ 1 \le j \le n]$$

$$\rho = \max_{i,e} \sum_j |\pi_{i,e,j}|$$

ITERATE: For $t = 1, 2, \ldots$

$$q_{t,i,e} = \sum_{k:e \in E_{i,k}} \frac{\exp(\boldsymbol{\lambda}_t \cdot \boldsymbol{\pi}_{i,e})}{1 + \sum_{e' \in E_{i,k}} \exp(\boldsymbol{\lambda}_t \cdot \boldsymbol{\pi}_{i,e'})} \qquad [1 \le i \le m,\ e \in E_i]$$

$$W_{t,j}^{+} = \sum_{i,e:\pi_{i,e,j}>0} \frac{q_{t,i,e}\, \pi_{i,e,j}}{s_i} \qquad W_{t,j}^{-} = \sum_{i,e:\pi_{i,e,j}<0} \frac{-q_{t,i,e}\, \pi_{i,e,j}}{s_i} \qquad [1 \le j \le n]$$

$$\Lambda_{t,j} = \frac{1}{2} \ln\left(\frac{W_{t,j}^{+}}{W_{t,j}^{-}}\right) \qquad [1 \le j \le n]$$

$$\boldsymbol{\lambda}_{t+1} = \boldsymbol{\lambda}_t - \frac{\boldsymbol{\Lambda}_t}{\rho}$$

Figure 3: A boosting based algorithm for generalized label ranking.

## 3 Minimizing the Generalized Ranking Error

Our goal is to minimize the generalized error for a given training set $S$ and graph decomposition procedure $\mathcal{A}$. This task generalizes standard classification problems which are known to be NP-complete. Hence we do not attempt to minimize the error directly but rather minimize a smooth, strictly convex, upper bound on $\varepsilon_{\text{gen}}$. The disagreement of $f(\mathbf{x}_i)$ and a preference graph $G_{i,k} = (V_{i,k}, E_{i,k})$ can be upper bounded by

$$\delta(f, \mathbf{x}_i, G_{i,k}) \le \log_2\left(1 + \sum_{e \in E_{i,k}} \exp\Big(f\big(\mathbf{x}_i, \text{term}(e)\big) - f\big(\mathbf{x}_i, \text{init}(e)\big)\Big)\right)$$

Denoting the right hand side of the above as $L(f(\mathbf{x}_i), G_{i,k})$, we define the *loss* attained by $f$ on the entire training set $S$ to be

$$\mathcal{L}(f, S, \mathcal{A}) = \sum_{i=1}^{m} \frac{1}{s_i} \sum_{k=1}^{s_i} L(f(\mathbf{x}_i), G_{i,k}) \quad \text{where } G_{i,1}, \ldots, G_{i,s_i} = \mathcal{A}(\mathbf{G}_i) \ .$$

From the definition of the generalized error in Eq. (1), we conclude the upper bound $\varepsilon_{\text{gen}}(f, S, \mathcal{A}) \le \mathcal{L}(f, S, \mathcal{A})$. A boosting-based algorithm that globally minimizes the loss is given in Fig. 3. On every iteration, a weight $q_{t,i,e}$ is calculated for every edge in the training data, and the algorithm focuses on satisfying each edge with proportion to its weight. This set of weights plays the role of the distribution vector common in boosting algorithms for classification. The following theorem bounds the decrease in loss on every iteration of the algorithm by a non-negative auxiliary function.

**Theorem 1** *Let $S = \{(\mathbf{x}_i, \mathbf{G}_i)\}_{i=1}^{m}$ be a training set such that every $\mathbf{x}_i \in \mathcal{X}$ and every $\mathbf{G}_i$ is a preference graph. Let $\mathcal{A}$ be a graph decomposition procedure that defines for each preference graph $\mathbf{G}_i$ a set of subgraphs $\{G_{i,1}, \ldots, G_{i,s_i}\} = \mathcal{A}(\mathbf{G}_i)$. Denote by $f_t$ the ranking function obtained at iteration $t$ of the algorithm given in Fig. 3 ($f_t = \sum_j \lambda_{t,j} h_j$). Using the notation defined in Fig. 3, the decrease in loss on iteration $t$ is bounded by*

$$\mathcal{L}(f_t, S, \mathcal{A}) - \mathcal{L}(f_{t+1}, S, \mathcal{A}) \ge \frac{1}{\rho} \sum_{j=1}^{n} \left(\sqrt{W_{t,j}^{+}} - \sqrt{W_{t,j}^{-}}\right)^2 \ .$$

**Proof** Define $\Delta_{t,i,k}$ to be the difference between the loss attained by $f_t$ and the loss attained by $f_{t+1}$ on $(\mathbf{x}_i, G_{i,k})$, that is $\Delta_{t,i,k} = L(f_t(\mathbf{x}_i), G_{i,k}) - L(f_{t+1}(\mathbf{x}_i), G_{i,k})$, and define $\phi_{t,i,k} = \sum_{e \in E_{i,k}} \exp(\boldsymbol{\lambda}_t \cdot \boldsymbol{\pi}_{i,e})$. We can now rewrite $L(f_t(\mathbf{x}_i), G_{i,k})$ as $\log(1 + \phi_{t,i,k})$. Using the inequality $-\log(1 - a) \geq a$ (which holds when $\log(1 - a)$ is defined), we get

$$
\begin{aligned}
\Delta_{t,i,k} &= \log\big(1 + \phi_{t,i,k}\big) - \log\big(1 + \phi_{t+1,i,k}\big) = -\log\left(1 - \frac{\phi_{t,i,k} - \phi_{t+1,i,k}}{1 + \phi_{t,i,k}}\right) \\
&\geq \frac{\phi_{t,i,k} - \phi_{t+1,i,k}}{1 + \phi_{t,i,k}} = \sum_{e \in E_{i,k}} \frac{\exp(\boldsymbol{\lambda}_t \cdot \boldsymbol{\pi}_{i,e}) - \exp(\boldsymbol{\lambda}_{t+1} \cdot \boldsymbol{\pi}_{i,e})}{1 + \sum_{e' \in E_{i,k}} \exp(\boldsymbol{\lambda}_t \cdot \boldsymbol{\pi}_{i,e'})} \, . \quad (3)
\end{aligned}
$$

The algorithm sets $\boldsymbol{\lambda}_{t+1} = \boldsymbol{\lambda}_t - (1/\rho)\boldsymbol{\Lambda}_t$ and therefore $\exp(\boldsymbol{\lambda}_{t+1} \cdot \boldsymbol{\pi}_{i,e})$ in Eq. (3) can be replaced by $\exp(\boldsymbol{\lambda}_t \cdot \boldsymbol{\pi}_{i,e}) \exp(-(1/\rho)\boldsymbol{\Lambda}_t \cdot \boldsymbol{\pi}_{i,e})$, yielding:

$$
\Delta_{t,i,k} \geq \sum_{e \in E_{i,k}} \left(\frac{\exp(\boldsymbol{\lambda}_t \cdot \boldsymbol{\pi}_{i,e})}{1 + \sum_{e' \in E_{i,k}} \exp(\boldsymbol{\lambda}_t \cdot \boldsymbol{\pi}_{i,e'})}\right)\left(1 - \exp\left(-\frac{1}{\rho}\boldsymbol{\Lambda}_t \cdot \boldsymbol{\pi}_{i,e}\right)\right) \, .
$$

Summing both sides of the above over the subgraphs in $\mathcal{A}(\mathbf{G}_i)$, and plugging in $q_{t,i,e}$,

$$
\begin{aligned}
\sum_{k=1}^{s_i} \Delta_{t,i,k} &\geq \sum_{e \in E_i} \left(\sum_{k:e \in E_{i,k}} \frac{\exp(\boldsymbol{\lambda}_t \cdot \boldsymbol{\pi}_{i,e})}{1 + \sum_{e' \in E_{i,k}} \exp(\boldsymbol{\lambda}_t \cdot \boldsymbol{\pi}_{i,e'})}\right)\left(1 - \exp\left(-\frac{1}{\rho}\boldsymbol{\Lambda}_t \cdot \boldsymbol{\pi}_{i,e}\right)\right) \\
&= \sum_{e \in E_i} q_{t,i,e}\left(1 - \exp\left(-\frac{1}{\rho}\boldsymbol{\Lambda}_t \cdot \boldsymbol{\pi}_{i,e}\right)\right) \, . \quad (4)
\end{aligned}
$$

We now rewrite $(1/\rho)\boldsymbol{\Lambda}_t \cdot \boldsymbol{\pi}_{i,e}$ in more convenient form

$$
-\frac{1}{\rho}\boldsymbol{\Lambda}_t \cdot \boldsymbol{\pi}_{i,e} = -\sum_{j=1}^{n} \frac{1}{\rho}\Lambda_{t,j}\pi_{i,e,j} = \sum_{j=1}^{n} (|\pi_{i,e,j}|/\rho)\,(-\text{sign}(\pi_{i,e,j})\Lambda_{t,j}) \, . \quad (5)
$$

The rationale behind this rewriting is that we now think of $(|\pi_{i,e,1}|/\rho), \ldots, (|\pi_{i,e,n}|/\rho)$ as coefficients in a subconvex combination of $(-\text{sign}(\pi_{i,e,1})\Lambda_{t,1}), \ldots, (-\text{sign}(\pi_{i,e,n})\Lambda_{t,n})$, since $\forall j \ (|\pi_{i,e,j}|/\rho) \geq 0$ and from the definition of $\rho$, $\sum_j (|\pi_{i,e,1}|/\rho) \leq 1$. Plugging Eq. (5) into Eq. (4) and using the concavity of the function $1 - \exp(\cdot)$ in Eq. (4), we obtain

$$
\begin{aligned}
\sum_{k=1}^{s_i} \Delta_{t,i,k} &\geq \sum_{e \in E_i} q_{t,i,e}\left(1 - \exp\left(\sum_{j=1}^{n} (|\pi_{i,e,j}|/\rho)\,(-\text{sign}(\pi_{i,e,j})\Lambda_{t,j})\right)\right) \\
&\geq \sum_{e \in E_{i,k}} \sum_{j=1}^{n} q_{t,i,e}(|\pi_{i,e,j}|/\rho)\big(1 - \exp\left(-\text{sign}(\pi_{i,e,j})\Lambda_{t,j}\right)\big) \, .
\end{aligned}
$$

Finally, we sum both sides of the above over all of $S$ and plug in $W^+$, $W^-$ and $\boldsymbol{\Lambda}$ to get

$$
\begin{aligned}
\mathcal{L}(f_t, S, \mathcal{A}) - \mathcal{L}(f_{t+1}, S, \mathcal{A}) &= \sum_{i=1}^{n} \sum_{k=1}^{s_i} \Delta_{t,i,k} \\
&\geq \frac{1}{\rho} \sum_{j=1}^{n} \sum_{i=1}^{m} \sum_{e \in E_i} \frac{q_{t,i,e}|\pi_{i,e,j}|}{s_i}\big(1 - \exp\left(-\text{sign}(\pi_{i,e,j})\Lambda_{t,j}\right)\big) \\
&= \frac{1}{\rho} \sum_{j=1}^{n} \left[W_{t,j}^+\left(1 - \frac{\sqrt{W_{t,j}^-}}{\sqrt{W_{t,j}^+}}\right) + W_{t,j}^-\left(1 - \frac{\sqrt{W_{t,j}^+}}{\sqrt{W_{t,j}^-}}\right)\right] \\
&= \frac{1}{\rho} \sum_{j=1}^{n} \left(\sqrt{W_{t,j}^+} - \sqrt{W_{t,j}^-}\right)^2 \, . \qquad\blacksquare
\end{aligned}
$$

Thm. 1 proves that the losses attained on each iteration form a monotonically non-increasing sequence of positive numbers, that must therefore converge. However, we are interested in proving a stronger claim, namely that the vector sequence $(\boldsymbol{\lambda}_t)_{t=1}^{\infty}$ converges to a globally optimal weight-vector $\boldsymbol{\lambda}^{\star}$. Since the loss is a convex function, it suffices to show that the vector sequence converges to a stationary point of the loss. It is easily verified that the non-negative auxiliary function which bounds the decrease in loss equals zero only at stationary points of the loss. This fact implies that $(\boldsymbol{\lambda}_t)_{t=1}^{\infty}$ indeed converges to $\boldsymbol{\lambda}^{\star}$ if the set of all feasible values for $\boldsymbol{\lambda}$ is compact and the loss has a unique global minimum. Compactness of the feasible set and uniqueness of the optimum can be explicitly enforced by adding a form of natural regularization to the boosting algorithm. The specifics of this technique exceed the scope of this paper and are discussed in [5]. In all, the boosting algorithm of Fig. 3 converges to the globally optimal weight-vector $\boldsymbol{\lambda}^{\star}$.

## 4 Experiments

To demonstrate our framework, we chose to learn a category ranking problem on a subset of the *Reuters Corpus, Vol. 1* [14]. The full Reuters corpus is comprised of approximately $800,000$ textual news articles, collected over a period of 12 months in 1996–1997. Most of the articles are labeled by one or more categories. For the purpose of these experiments, we limited ourselves to the subset of articles collected during January 1997: approximately $66,000$ articles labeled by 103 different categories.

|        | $\varepsilon_{0-1}$ | $\varepsilon_{\text{dis}}$ | $\varepsilon_{\text{Dom}}$ | $\varepsilon_{\text{dom}}$ |
|--------|-------|-------|------|------|
| $0-1$  | 0.63  | 0.068 | 0.42 | 0.12 |
| dis    | 0.73  | 0.063 | 0.51 | 0.14 |
| Dom    | 0.59  | 0.049 | 0.35 | 0.10 |
| dom    | 0.59  | 0.067 | 0.41 | 0.10 |

Figure 4: The test error averaged over 5-fold cross validation. The rows correspond to different optimization problems: minimizing $\varepsilon_{0-1}$, $\varepsilon_{\text{dis}}$, $\varepsilon_{\text{Dom}}$ and $\varepsilon_{\text{dom}}$. Errors are measured using all 4 error measures.

An interesting aspect of the Reuters corpus is that the categories are arranged in a hierarchy. The set of possible labels contains both general categories and more specific ones, where the specific categories refine the general categories. This concept is best explained with an example: three of the categories in the corpus are *Economics*, *Government Finance* and *Government Borrowing*. It would certainly be correct to categorize an article on *government borrowing* as either *government finance* or *economics*, however these general categories are less specific and do not describe the article as well. Furthermore, misclassifying such an article as *government revenue* is by far better than misclassifying it as *sports*. In summary, the category hierarchy induces a preference over the set of labels. We exploit this property to generate supervision for the label ranking problem at hand.

Formally, we view every category as a vertex in a rooted tree, where the tree root corresponds to a general abstract category that is relevant to all of the articles in the corpus and every category is a specific instance of its parent in the tree. The labels associated with an article constitute a set of paths from the tree root to a set of leaves. The original corpus is somewhat inconsistent in that not all paths end in a leaf, but rather end in some inner vertex. To fix this inconsistency, we added a dummy child vertex to every inner vertex and diverted all paths that originally end in this inner vertex to its new child. Our learning problem then becomes the problem of ranking *leaves*. The severity of wrongly categorizing an article by a leaf is proportional to the graph distance between this leaf and the closest correct leaf given in the corpus. The preference graph that encodes this preference is a multi-layer graph where the top layer contains all of the correct labels, the second layer contains all of their sibling vertices in the tree and so on. Every vertex in the multi-layer preference graph has outgoing edges to all vertices in lower layers, but there are no edges between vertices in the same layer. For practical purposes, we conducted experiments using only 3-layer preference graphs generated by collapsing all of the layers below 3 to a single layer.

All of the experiments were carried out using 5-fold cross validation. The word counts for each article were used to construct base ranking functions in the following way: for every word $w$ and every category $y$, let $w(x_i)$ denote the number of appearances of $w$ in the article $x_i$. Then, define

$$h_{w,y}(x_i, y_i) = \begin{cases} \log(w(x_i)) + 1 & \text{if } w(x_i) > 0 \text{ and } y_i = y \\ 0 & \text{otherwise .} \end{cases} \tag{6}$$

For each training set, we first applied a heuristic feature selection method common in boosting applications [10] to select some 3200 informative words. These words then define $103 \cdot 3200$ base ranking functions as shown in Eq. (6). Next, we ran our learning algorithm using each of the 4 graph decomposition procedures discussed above: zero-one, disagreement, domination and dominated. After learning each problem, we calculated all four error measures on the test data. The results are presented in Fig. 4. Two points are worth noting. First, these results are not comparable with previous results for multilabel problems using this corpus, since label ranking is a more difficult task. For instance, an average preference graph in the test data has 820 edges, and the error for such a graph equals zero only if every single edge agrees with the ranking function. Second, the experiments clearly indicate that the results obtained by minimizing the domination loss are better than the other ranking losses, no matter what error is used for evaluation. In particular, employing the domination loss yields significantly better results than using the disagreement loss which has been the commonly used decomposition method in categorization problems [7, 10, 6, 4].

## 5  Summary

We presented a general framework for label ranking problems by means of preference graphs and the graph decomposition procedure. This framework was shown to generalize other decision problems, most notably multilabel categorization. We then described and analyzed a boosting algorithm that works with any choice of graph decomposition. We are currently exporting the approach to learning in inner product spaces, where different graph decomposition procedures result in different bindings of slack variables. Another interesting question is whether the graph decomposition approach can be combined with probabilistic models for orderings [9] to achieve algorithmic efficiency.

## References

[1] M. Collins and N. Duffy. New ranking algorithms for parsing and tagging: Kernels over discrete structures, and the voted perceptron. In *30th Annual Meeting of the ACL*, 2002.

[2] M. Collins, R.E. Schapire, and Y. Singer. Logistic regression, AdaBoost and Bregman distances. *Machine Learning*, 47(2/3):253–285, 2002.

[3] K. Crammer and Y. Singer. Pranking with ranking. NIPS 14, 2001.

[4] K. Crammer and Y. Singer. A new family of online algorithms for category ranking. *Jornal of Machine Learning Research*, 3:1025–1058, 2003.

[5] O. Dekel, S. Shalev-Shwartz, and Y. Singer. Smooth epsilon-insensitive regression by loss symmetrization. COLT 16, 2003.

[6] A. Elisseeff and J. Weston. A kernel method for multi-labeled classification. NIPS 14, 2001.

[7] Y. Freund, R. Iyer, R. E.Schapire, and Y. Singer. An effi cient boosting algorithm for combining preferences. In *Machine Learning: Proc. of the Fifteenth International Conference*, 1998.

[8] G. Lebanon and J. Lafferty. Boosting and ML for exponential models. NIPS 14, 2001.

[9] G. Lebanon and J. Lafferty. Conditional models on the ranking poset. NIPS 15, 2002.

[10] R. E. Schapire and Y. Singer. BoosTexter: A boosting-based system for text categorization. *Machine Learning*, 32(2/3), 2000.

[11] A. Shashua and A. Levin. Ranking with large margin principle. NIPS 15, 2002.

[12] K. Toutanova and C. D. Manning. Feature selection for a rich HPSG grammar using decision trees. In *Proceedings of the Sixth Conference on Natural Language Learning (CoNLL)*, 2002.

[13] The Penn Treebank Project. http://www.cis.upenn.edu/~treebank/.

[14] Reuters Corpus Vol. 1. http://about.reuters.com/researchandstandards/corpus/.
